# Finite-Sample Convergence Rates for Q-Learning and Indirect Algorithms

**Michael Kearns and Satinder Singh**
AT&T Labs
180 Park Avenue
Florham Park, NJ 07932
{mkearns,baveja}@research.att.com

## Abstract

In this paper, we address two issues of long-standing interest in the reinforcement learning literature. First, what kinds of performance guarantees can be made for Q-learning after only a finite number of actions? Second, what quantitative comparisons can be made between Q-learning and model-based (indirect) approaches, which use experience to estimate next-state distributions for off-line value iteration?

We first show that both Q-learning and the indirect approach enjoy rather rapid convergence to the optimal policy as a function of the number of state transitions observed. In particular, on the order of only $(N \log(1/\epsilon)/\epsilon^2)(\log(N) + \log \log(1/\epsilon))$ transitions are sufficient for both algorithms to come within $\epsilon$ of the optimal policy, in an idealized model that assumes the observed transitions are "well-mixed" throughout an $N$-state MDP. Thus, the two approaches have roughly the same sample complexity. Perhaps surprisingly, this sample complexity is far less than what is required for the model-based approach to actually construct a good approximation to the next-state distribution. The result also shows that the amount of memory required by the model-based approach is closer to $N$ than to $N^2$.

For either approach, to remove the assumption that the observed transitions are well-mixed, we consider a model in which the transitions are determined by a fixed, arbitrary exploration policy. Bounds on the number of transitions required in order to achieve a desired level of performance are then related to the stationary distribution and mixing time of this policy.

## 1 Introduction

There are at least two different approaches to learning in Markov decision processes: *indirect* approaches, which use control experience (observed transitions and payoffs) to estimate a model, and then apply dynamic programming to compute policies from the estimated model; and *direct* approaches such as Q-learning [2], which use control

experience to directly learn policies (through value functions) without ever explicitly estimating a model. Both are known to converge *asymptotically* to the optimal policy [1, 3]. However, little is known about the performance of these two approaches after only a finite amount of experience.

A common argument offered by proponents of direct methods is that it may require much more experience to learn an accurate model than to simply learn a good policy. This argument is predicated on the seemingly reasonable assumption that an indirect method must first learn an accurate model in order to compute a good policy. On the other hand, proponents of indirect methods argue that such methods can do unlimited off-line computation on the estimated model, which may give an advantage over direct methods, at least if the model is accurate. Learning a good model may also be useful across tasks, permitting the computation of good policies for multiple reward functions [4]. To date, these arguments have lacked a formal framework for analysis and verification.

In this paper, we provide such a framework, and use it to derive the first finite-time convergence rates (sample size bounds) for both Q-learning and the standard indirect algorithm. An important aspect of our analysis is that we *separate* the quality of the policy generating experience from the quality of the two learning algorithms. In addition to demonstrating that both methods enjoy rather rapid convergence to the optimal policy as a function of the amount of control experience, the convergence rates have a number of specific and perhaps surprising implications for the hypothetical differences between the two approaches outlined above. Some of these implications, as well as the rates of convergence we derive, were briefly mentioned in the abstract; in the interests of brevity, we will not repeat them here, but instead proceed directly into the technical material.

## 2  MDP Basics

Let $M$ be an unknown $N$-state MDP with $A$ actions. We use $P^a_M(ij)$ to denote the probability of going to state $j$, given that we are in state $i$ and execute action $a$; and $R^a_M(i)$ to denote the reward received for executing $a$ from $i$ (which we assume is fixed and bounded between 0 and 1 without loss of generality). A policy $\pi$ assigns an action to each state. The value of state $i$ under policy $\pi$, $V^\pi_M(i)$, is the expected discounted sum of rewards received upon starting in state $i$ and executing $\pi$ forever: $V^\pi_M(i) = \mathbf{E}_\pi[r_1 + \gamma r_2 + \gamma^2 r_3 + \cdots]$, where $r_t$ is the reward received at time step $t$ under a random walk governed by $\pi$ from start state $i$, and $0 \leq \gamma < 1$ is the discount factor. It is also convenient to define values for state-action pairs $(i, a)$: $Q^\pi_M(i, a) = R^a_M(i) + \gamma \sum_j P^a_M(ij) V^\pi_M(j)$. The goal of learning is to approximate the optimal policy $\pi^*$ that maximizes the value at every state; the optimal value function is denoted $Q^*_M$. Given $Q^*_M$, we can compute the optimal policy as $\pi^*(i) = \mathrm{argmax}_a\{Q^*_M(i, a)\}$.

If $M$ is given, *value iteration* can be used to compute a good approximation to the optimal value function. Setting our initial guess as $Q_0(i, a) = 0$ for all $(i, a)$, we iterate as follows:

$$Q_{\ell+1}(i, a) \;=\; R^a_M(i) + \gamma \sum_j [P^a_M(ij) V_\ell(j)] \tag{1}$$

where we define $V_\ell(j) = \max_b\{Q_\ell(j, b)\}$. It can be shown that after $\ell$ iterations, $\max_{(i,a)}\{|Q_\ell(i, a) - Q^*_M(i, a)|\} \leq \gamma^\ell$. Given any approximation $Q$ to $Q^*_M$ we can compute the greedy approximation $\pi$ to the optimal policy $\pi^*$ as $\pi(i) = \mathrm{argmax}_a\{Q(i, a)\}$.

## 3   The Parallel Sampling Model

In reinforcement learning, the transition probabilities $P_M^a(ij)$ are not given, and a good policy must be *learned* on the basis of observed experience (transitions) in $M$. Classical convergence results for algorithms such as Q-learning [1] implicitly assume that the observed experience is generated by an *arbitrary* "exploration policy" $\pi$, and then proceed to prove convergence to the optimal policy *if* $\pi$ meets certain minimal conditions — namely, $\pi$ must try every state-action pair infinitely often, with probability 1. This approach conflates two distinct issues: the quality of the exploration policy $\pi$, and the quality of reinforcement learning algorithms using experience generated by $\pi$. In contrast, we choose to separate these issues. If the exploration policy never or only very rarely visits some state-action pair, we would like to have this reflected as a factor in our bounds that depends *only on* $\pi$; a separate factor depending *only on the learning algorithm* will in turn reflect how efficiently a particular learning algorithm uses the experience generated by $\pi$. Thus, for a fixed $\pi$, all learning algorithms are placed on equal footing, and can be directly compared.

There are probably various ways in which this separation can be accomplished; we now introduce one that is particularly clean and simple. We would like a model of the *ideal* exploration policy — one that produces experiences that are "well-mixed", in the sense that every state-action pair is tried with equal frequency. Thus, let us define a *parallel sampling* subroutine PS($M$) that behaves as follows: a *single* call to PS($M$) returns, for *every* state-action pair $(i, a)$, a random next state $j$ distributed according to $P_M^a(ij)$. Thus, every state-action pair is executed simultaneously, and the resulting $N \times A$ next states are reported. A single call to PS($M$) is therefore really simulating $N \times A$ transitions in $M$, and we must be careful to multiply the number of calls to PS($M$) by this factor if we wish to count the *total* number of transitions witnessed.

What is PS($M$) modeling? It is modeling the idealized exploration policy that manages to visit every state-action pair in succession, without duplication, and without fail. It should be intuitively obvious that such an exploration policy would be optimal, from the viewpoint of gathering experience everywhere as rapidly as possible.

We shall first provide an analysis, in Section 5, of both direct and indirect reinforcement learning algorithms, in a setting in which the observed experience is generated by calls to PS($M$). Of course, in any given MDP $M$, there may not be *any* exploration policy that meets the ideal captured by PS($M$) — for instance, there may simply be some states that are very difficult for any policy to reach, and thus the experience generated by any policy will certainly not be equally mixed around the entire MDP. (Indeed, a call to PS($M$) will typically return a set of transitions that does not even correspond to a trajectory in $M$.) Furthermore, even if PS($M$) could be simulated by *some* exploration policy, we would like to provide more general results that express the amount of experience required for reinforcement learning algorithms under *any* exploration policy (where the amount of experience will, of course, depend on properties of the exploration policy).

Thus, in Section 6, we sketch how one can bound the amount of experience required under any $\pi$ in order to simulate calls to PS($M$). (More detail will be provided in a longer version of this paper.) The bound depends on natural properties of $\pi$, such as its stationary distribution and mixing time. Combined with the results of Section 5, we get the desired two-factor bounds discussed above: for both the direct and indirect approaches, a bound on the total number of transitions required, consisting of one factor that depends only on the algorithm, and another factor that depends only on the exploration policy.

# 4   The Learning Algorithms

We now explicitly state the two reinforcement learning algorithms we shall analyze and compare. In keeping with the separation between algorithms and exploration policies already discussed, we will phrase these algorithms in the parallel sampling framework, and Section 6 indicates how they generalize to the case of arbitrary exploration policies. We begin with the direct approach.

Rather than directly studying standard Q-learning, we will here instead examine a variant that is slightly easier to analyze, and is called *phased* Q-learning. However, we emphasize that *all of our results can be generalized to apply to standard Q-learning* (with learning rate $\alpha(i,a) = \frac{1}{t(i,a)}$, where $t(i,a)$ is the number of trials of $(i,a)$ so far). Basically, rather than updating the value function with *every* observed transition from $(i,a)$, phased Q-learning estimates the expected value of the next state from $(i,a)$ on the basis of *many* transitions, and only then makes an update. The memory requirements for phased Q-learning are essentially the same as those for standard Q-learning.

**Direct Algorithm — Phased Q-Learning:** As suggested by the name, the algorithm operates in phases. In each phase, the algorithm will make $m_D$ calls to $\text{PS}(M)$ (where $m_D$ will be determined by the analysis), thus gathering $m_D$ trials of every state-action pair $(i,a)$. At the $\ell$th phase, the algorithm updates the estimated value function as follows: for every $(i,a)$,

$$\widehat{Q}_{\ell+1}(i,a) = R_M^a(i) + \gamma \frac{1}{m_D} \sum_{k=1}^{m_D} \widehat{V}_\ell(j_k^\ell) \qquad (2)$$

where $j_1^\ell, \ldots, j_{m_D}^\ell$ are the $m_D$ next states observed from $(i,a)$ on the $m_D$ calls to $\text{PS}(M)$ during the $\ell$th phase. The policy computed by the algorithm is then the greedy policy determined by the final value function. Note that phased Q-learning is quite like standard Q-learning, except that we gather statistics (the summation in Equation (2)) before making an update.

We now proceed to describe the standard indirect approach.

**Indirect Algorithm:** The algorithm first makes $m_I$ calls to $\text{PS}(M)$ to obtain $m_I$ next state samples for each $(i,a)$. It then builds an empirical model of the transition probabilities as follows: $\widehat{P}_M^a(ij) = \frac{\#(i \to_a j)}{m_I}$, where $\#(i \to_a j)$ is the number of times state $j$ was reached on the $m_I$ trials of $(i,a)$. The algorithm then does value iteration (as described in Section 2) on the fixed model $\widehat{P}_M^a(ij)$ for $\ell_I$ phases. Again, the policy computed by the algorithm is the greedy policy dictated by the final value function.

Thus, in phased Q-learning, the algorithm runs for some number $\ell_D$ phases, and *each phase* requires $m_D$ calls to $\text{PS}(M)$, for a total number of transitions $\ell_D \times m_D \times N \times A$. The direct algorithm first makes $m_I$ calls to $\text{PS}(M)$, and then runs $\ell_I$ phases of value iteration (which requires no additional data), for a total number of transitions $m_I \times N \times A$. The question we now address is: how large must $m_D, m_I, \ell_D, \ell_I$ be so that, with probability at least $1 - \delta$, the resulting policies have expected return within $\epsilon$ of the optimal policy in $M$? The answers we give yield perhaps surprisingly similar bounds on the total number of transitions required for the two approaches in the parallel sampling model.

# 5   Bounds on the Number of Transitions

We now state our main result.

**Theorem 1** *For any MDP M:*

- *For an appropriate choice of the parameters $m_I$ and and $\ell_I$, the total number of calls to PS($M$) required by the indirect algorithm in order to ensure that, with probability at least $1 - \delta$, the expected return of the resulting policy will be within $\epsilon$ of the optimal policy, is*

$$O((1/\epsilon^2)(\log(N/\delta) + \log\log(1/\epsilon)). \tag{3}$$

- *For an appropriate choice of the parameters $m_D$ and $\ell_D$, the total number of calls to PS($M$) required by phased Q-learning in order to ensure that, with probability at least $1 - \delta$, the expected return of the resulting policy will be within $\epsilon$ of the optimal policy, is*

$$O((\log(1/\epsilon)/\epsilon^2)(\log(N/\delta) + \log\log(1/\epsilon)). \tag{4}$$

*The bound for phased Q-learning is thus only $O(\log(1/\epsilon))$ larger than that for the indirect algorithm. Bounds on the total number of transitions witnessed in either case are obtained by multiplying the given bounds by $N \times A$.*

Before sketching some of the ideas behind the proof of this result, we first discuss some of its implications for the debate on direct versus indirect approaches. First of all, for *both* approaches, convergence is rather fast: with a total number of transitions only on the order of $N\log(N)$ (fixing $\epsilon$ and $\delta$ for simplicity), near-optimal policies are obtained. This represents a considerable advance over the classical asymptotic results: instead of saying that an infinite number of visits to every state-action pair are required to converge to the optimal policy, we are claiming that a rather small number of visits are required to get close to the optimal policy. Second, by our analysis, the two approaches have similar complexities, with the number of transitions required differing by only a $\log(1/\epsilon)$ factor in favor of the indirect algorithm. Third — and perhaps surprisingly — note that since only $O(\log(N))$ calls are being made to PS($M$) (again fixing $\epsilon$ and $\delta$), and since the number of trials *per state-action pair* is exactly the number of calls to PS($M$), the total number of non-zero entries in the model $\widehat{P}_M^a(ij)$ built by the indirect approach is in fact only $O(\log(N))$. In other words, $\widehat{P}_M^a(ij)$ will be extremely sparse — and thus, a terrible approximation to the true transition probabilities — yet still good enough to derive a near-optimal policy! Clever representation of $\widehat{P}_M^a(ij)$ will thus result in total memory requirements that are only $O(N\log(N))$ rather than $O(N^2)$. Fourth, although we do not have space to provide any details, if instead of a single reward function, we are provided with $L$ reward functions (where the $L$ reward functions are given in advance of observing any experience), then for both algorithms, the number of transitions required to compute near-optimal policies for all $L$ reward functions simultaneously is only a factor of $O(\log(L))$ greater than the bounds given above.

Our own view of the result and its implications is:

- Both algorithms enjoy rapid convergence to the optimal policy as a function of the amount of experience.

- In general, neither approach enjoys a significant advantage in convergence rate, memory requirements, or handling multiple reward functions. Both are quite efficient on all counts.

We do not have space to provide a detailed proof of Theorem 1, but instead provide some highlights of the main ideas. The proofs for both the indirect algorithm and phased Q-learning are actually quite similar, and have at their heart two slightly

different *uniform convergence* lemmas. For phased Q-learning, it is possible to show that, for any bound $\ell_D$ on the number of phases to be executed, and for any $\tau > 0$, we can choose $m_D$ so that

$$\left| (1/m_D) \sum_{k=1}^{m_D} \widehat{V}_\ell(j_k^\ell) - \sum_j P_{ij}^a \widehat{V}_\ell(j) \right| \leq \tau \tag{5}$$

will hold *simultaneously* for *every* $(i, a)$ and for *every* phase $\ell = 1, \ldots, \ell_D$. In other words, at the end of every phase, the empirical estimate of the expected next-state value for every $(i, a)$ will be close to the true expectation, where here the expectation is with respect to the current estimated value function $\widehat{V}_\ell$.

For the indirect algorithm, a slightly more subtle uniform convergence argument is required. Here we show that it is possible to choose, for any bound $\ell_I$ on the number of iterations of value iteration to be executed on the $\widehat{P}_M^a(ij)$, and for any $\tau > 0$, a value $m_I$ such that

$$\left| \sum_j \widehat{P}_{ij}^a V_\ell(j) - \sum_j P_{ij}^a V_\ell(j) \right| \leq \tau \tag{6}$$

for every $(i, a)$ and every phase $\ell = 1, \ldots, \ell_I$, where the $V_\ell(j)$ are the value functions resulting from performing *true* value iteration (that is, on the $P_M^a(ij)$). Equation (6) essentially says that expectations of the true value functions are quite similar under either the true or estimated model, even though the indirect algorithm never has access to the true value functions.

In either case, the uniform convergence results allow us to argue that the corresponding algorithms still achieve successive *contractions*, as in the classical proof of value iteration. For instance, in the case of phased Q-learning, if we define $\Delta_\ell = \max_{(i,a)}\{|\widehat{Q}_\ell(i, a) - Q_\ell(i, a)|\}$, we can derive a recurrence relation for $\Delta_{\ell+1}$ as follows:

$$|\widehat{Q}_{\ell+1}(i, a) - Q_{\ell+1}(i, a)| = \left| \gamma(1/m) \sum_{k=1}^{m} \widehat{V}_\ell(j_k^\ell) - \gamma \sum_j P_{ij}^a V_\ell(j) \right| \tag{7}$$

$$\leq \gamma \max_{\alpha \in \{\tau, -\tau\}} \left\{ \left| \left( \sum_j P_{ij}^a \widehat{V}_\ell(j) + \alpha \right) - \sum_j P_{ij}^a V_\ell(j) \right| \right\} \tag{8}$$

$$\leq \gamma\tau + \gamma\Delta_\ell. \tag{9}$$

Here we have made use of Equation (5). Since $\Delta_0 = 0$ ($\widehat{Q}_0 = Q_0$), this recurrence gives $\Delta_\ell \leq \tau(\gamma/(1-\gamma))$ for any $\ell$. From this it is not hard to show that for any $(i, a)$

$$|\widehat{Q}_\ell(i, a) - Q^*(i, a)| \leq \tau(\gamma/(1 - \gamma)) + \gamma^\ell. \tag{10}$$

From this it can be shown that the regret in expected return suffered by the policy computed by phased Q-Learning after $\ell$ phases is at most $(\tau\gamma/(1-\gamma) + \gamma^\ell)(2/(1-\gamma))$. The proof proceeds by setting this regret smaller than the desired $\epsilon$, solving for $\ell$ and $\tau$, and obtaining the resulting bound on $m_D$. The derivation of bounds for the indirect algorithm is similar.

## 6  Handling General Exploration Policies

As promised, we conclude our technical results by briefly sketching how we can translate the bounds obtained in Section 5 under the idealized parallel sampling model into

bounds applicable when *any* fixed policy $\pi$ is guiding the exploration. Such bounds must, of course, depend on properties of $\pi$. Due to space limitations, we can only outline the main ideas; the formal statements and proofs are deferred to a longer version of the paper.

Let us assume for simplicity that $\pi$ (which may be a stochastic policy) defines an *ergodic* Markov process in the MDP $M$. Thus, $\pi$ induces a unique *stationary distribution* $P_{M,\pi}(i,a)$ over state-action pairs — intuitively, $P_{M,\pi}(i,a)$ is the frequency of executing action $a$ from state $i$ during an infinite random walk in $M$ according to $\pi$. Furthermore, we can introduce the standard notion of the *mixing time* of $\pi$ to its stationary distribution — informally, this is the number $T_\pi$ of steps required such that the distribution induced on state-action pairs by $T_\pi$-step walks according to $\pi$ will be "very close" to $P_{M,\pi}$ [1]. Finally, let us define $\rho_\pi = \min_{(i,a)}\{P_{M,\pi}(i,a)\}$.

Armed with these notions, it is not difficult to show that the number of steps we must take under $\pi$ in order to simulate, with high probability, a call to the oracle $PS(M)$, is polynomial in the quantity $T_\pi/\rho_\pi$. The intuition is straightforward: at most every $T_\pi$ steps, we obtain an "almost independent" draw from $P_{M,\pi}(i,a)$; and with each independent draw, we have at least probability $\rho$ of drawing any particular $(i,a)$ pair. Once we have sampled every $(i,a)$ pair, we have simulated a call to $PS(M)$. The formalization of these intuitions leads to a version of Theorem 1 applicable to any $\pi$, in which the bound is multiplied by a factor polynomial in $T_\pi/\rho_\pi$, as desired.

However, a better result is possible. In cases where $\rho_\pi$ may be small or even 0 (which would occur when $\pi$ simply does not ever execute some action from some state), the factor $T_\pi/\rho_\pi$ is large or infinite and our bounds become weak or vacuous. In such cases, it is better to define the sub-MDP $M_\pi(\alpha)$, which is obtained from $M$ by simply deleting any $(i,a)$ for which $P_{M,\pi}(i,a) < \alpha$, where $\alpha > 0$ is a parameter of our choosing. In $M_\pi(\alpha)$, $\rho_\pi > \alpha$ by construction, and we may now obtain convergence rates to the optimal policy in $M_\pi(\alpha)$ for both Q-learning and the indirect approach like those given in Theorem 1, multiplied by a factor polynomial in $T_\pi/\alpha$. (Technically, we must slightly alter the algorithms to have an initial phase that detects and eliminates small-probability state-action pairs, but this is a minor detail.) By allowing $\alpha$ to become smaller as the amount of experience we receive from $\pi$ grows, we can obtain an "anytime" result, since the sub-MDP $M_\pi(\alpha)$ approaches the full MDP $M$ as $\alpha \to 0$.

## Footnotes

[1] Formally, the degree of closeness is measured by the distance between the transient and stationary distributions. For brevity here we will simply assume this parameter is set to a very small, constant value.

# References

[1] Jaakkola, T., Jordan, M. I., Singh, S. On the convergence of stochastic iterative dynamic programming algorithms. *Neural Computation*, 6(6), 1185–1201, 1994.

[2] C. J. C. H. Watkins. Learning from Delayed Rewards. Ph.D. thesis, Cambridge University, 1989.

[3] R. S. Sutton and A. G. Barto. *Reinforcement Learning: An Introduction.* MIT Press, 1998.

[4] S. Mahadevan. Enhancing Transfer in Reinforcement Learning by Building Stochastic Models of Robot Actions. In *Machine Learning: Proceedings of the Ninth International Conference*, 1992.

